# Information-geometric decomposition in spike analysis

**Hiroyuki Nakahara**[*], **Shun-ichi Amari**
Lab. for Mathematical Neuroscience, RIKEN Brain Science Institute
2-1 Hirosawa, Wako, Saitama, 351-0198 Japan
{*hiro, amari*}*@brain.riken.go.jp*

## Abstract

We present an information-geometric measure to systematically investigate neuronal firing patterns, taking account not only of the second-order but also of higher-order interactions. We begin with the case of two neurons for illustration and show how to test whether or not any pairwise correlation in one period is significantly different from that in the other period. In order to test such a hypothesis of different firing rates, the correlation term needs to be singled out 'orthogonally' to the firing rates, where the null hypothesis might not be of independent firing. This method is also shown to directly associate neural firing with behavior via their mutual information, which is decomposed into two types of information, conveyed by mean firing rate and coincident firing, respectively. Then, we show that these results, using the 'orthogonal' decomposition, are naturally extended to the case of three neurons and $n$ neurons in general.

## 1 Introduction

Based on the theory of hierarchical structure and related invariant decomposition of interactions by information geometry [3], the present paper briefly summarizes methods useful for systematically analyzing a population of neural firing [9].

Many researches have shown that the mean firing rate of a single neuron may carry significant information on sensory and motion signals. Information conveyed by populational firing, however, may not be only an accumulation of mean firing rates. Other statistical structure, e.g., coincident firing [13, 14], may also carry behavioral information. One obvious step to investigate this issue is to single out a contribution by coincident firing between two neurons, i.e., the pairwise correlation [2, 6].

In general, however, it is not sufficient to test a pairwise correlation of neural firing, because there can be triplewise and higher correlations. For example, three variables (neurons) are not independent in general even when they are pairwise independent.

We need to establish a systematic method of analysis, including these higher-order

---
[*] also affiliated with Dept. of Knowledge Sci., Japan Advanced Inst. of Sci. & Tech.

correlations [1, 5, 7, 13]. We propose one approach, the information-geometric measure that uses the dual orthogonality of the natural and expectation parameters in exponential family distributions [4]. We represent a neural firing pattern by a binary random vector $\boldsymbol{x}$. The probability distribution of firing patterns can be expanded by a log linear model, where the set $\{p(\boldsymbol{x})\}$ of all the probability distributions forms a $(2^n - 1)$-dimensional manifold $\boldsymbol{S}_n$. Each $p(\boldsymbol{x})$ is given by $2^n$ probabilities

$$p_{i_1 \cdots i_n} = \mathrm{Prob}\{X_1 = i_1, \cdots, X_n = i_n\}, \quad i_k = 0, 1, \quad \text{subject to} \quad \sum_{i_1, \cdots, i_n} p_{i_1 \cdots i_n} = 1$$

and expansion in $\log p(\boldsymbol{x})$ is given by

$$\log p(\boldsymbol{x}) = \sum \theta_i x_i + \sum_{i<j} \theta_{ij} x_i x_j + \sum_{i<j<k} \theta_{ijk} x_i x_j x_k \cdots + \theta_{1\ldots n} x_1 \cdots x_n - \psi,$$

where indices of $\theta_{ijk}$, etc. satisfy $i < j < k$, etc. We can have a general theory of this $n$ neuron case [3, 9], however, to be concrete given the limited space, we mainly discuss two and three neuron cases in the present paper. Our method shares some features with previous studies (e.g. [7]) in use of the log linear model. Yet, we make explicit use of the dual orthogonality so that the method becomes more transparent and more systematic.

In the present paper, we are interested in addressing two issues: (1) to analyze correlated firing of neurons and (2) to connect such a technique with behavioral events. In (1), previous studies often assumed independent firing as the null hypothesis. However, for example, when we compare firing patterns in two periods, as control and 'test' periods, there may exist a weak correlation in the control period. Hence, benefiting from the 'orthogonal' coordinates, we develop a method applicable to the null hypothesis of non-independent firing, irrespective of firing rates. It is equally important to relate such a method with investigation of behavioral significance as (2). We show that we can do so, using orthogonal decomposition of the mutual information (MI) between firing and behavior [11, 12].

In the following, we discuss first the case of two neurons and then the case of three neurons, demonstrating our method with artificial simulated data. The validity of our method has been shown also with experimental data[9, 10] but not shown here due to the limited space.

## 2  Information-geometric measure: case of two neurons

We denote two neurons by $X_1$ and $X_2$ ($X_i = 1, 0$ indicates if neuron $i$ has a spike or not in a short time bin). Its joint probability $p(\boldsymbol{x})$, $\boldsymbol{x} = (x_1, x_2)$, is given by $p_{ij} = \mathrm{Prob}\{x_1 = i; x_2 = j\} > 0$, $i, j = 0, 1$. Among four probabilities, $\{p_{00}, p_{01}, p_{10}, p_{11}\}$, only three are free. The set of all such distributions of $\boldsymbol{x}$ forms a three-dimensional manifold $\boldsymbol{S}_2$. Any three of $p_{ij}$ can be used as a coordinate system of $\boldsymbol{S}_2$.

There are many different coordinate systems of $\boldsymbol{S}_2$. The coordinates of the expectation parameters, called $\eta$-coordinates, $\boldsymbol{\eta} = (\eta_1, \eta_2, \eta_{12})$, is given by

$$\eta_i = \mathrm{Prob}\{x_i = 1\} = E[x_i], \quad i = 1, 2, \quad \eta_3 = \eta_{12} = E[x_1 x_2] = p_{12},$$

where $E$ denotes the expectation and $\eta_i$ and $\eta_{12}$ correspond to the mean firing rates and the mean coincident firing, respectively.

As other coordinate systems, we can also use the triplet, $(\eta_1, \eta_2, \mathrm{Cov}[X_1, X_2])$, where $\mathrm{Cov}[X_1, X_2]$ is the covariance, and/or the triplet $(\eta_1, \eta_2, \rho)$, where $\rho$ is the correlation coefficient (COR), $\rho = \frac{\eta_{12} - \eta_1 \eta_2}{\sqrt{\eta_1(1-\eta_1)\eta_2(1-\eta_2)}}$, often called N-JPSTH [2].

Which quantity would be convenient to represent the pairwise correlational component? It is desirable to define the degree of the correlation independently from the marginals $(\eta_1, \eta_2)$. To this end, we use the 'orthogonal' coordinates $(\eta_1, \eta_2, \theta)$, originating from information geometry of $\boldsymbol{S}_2$, so that the coordinate curve of $\theta$ is always orthogonal to those of $\eta_1$ and $\eta_2$.

The orthogonality of two directions in $\boldsymbol{S}_2$ ($\boldsymbol{S}_n$ in general) is defined by the Riemannian metric due to the Fisher information matrix [8, 4]. Denoting any coordinates in $\boldsymbol{S}_n$ by $\boldsymbol{\xi} = (\xi_1, ..., \xi_n)$, the Fisher information matrix $G$ is given by

$$G = (g_{ij}), \quad g_{ij}(\boldsymbol{\xi}) = E\left[\frac{\partial}{\partial \xi_i} l(\boldsymbol{x}; \boldsymbol{\xi}) \frac{\partial}{\partial \xi_j} l(\boldsymbol{x}; \boldsymbol{\xi})\right]. \tag{1}$$

where $l(\boldsymbol{x}; \boldsymbol{\xi}) = \log p(\boldsymbol{x}; \boldsymbol{\xi})$. The orthogonality between $\xi_i$ and $\xi_j$ is defined by $g_{ij}(\boldsymbol{\xi}) = 0$. In case of $\boldsymbol{S}_2$, we desire to have $E\left[\frac{\partial}{\partial \theta} l(\boldsymbol{x}; \eta_1, \eta_2, \theta) \frac{\partial}{\partial \eta_i} l(\boldsymbol{x}; \eta_1, \eta_2, \theta)\right] = 0$ $(i = 1, 2)$. When $\theta$ is orthogonal to $(\eta_1, \eta_2)$, we say that $\theta$ represents pure correlations independently of marginals. Such $\theta$ is given by the following theorem.

**Theorem 1.**    The coordinate

$$\theta = \log \frac{p_{11} p_{00}}{p_{01} p_{10}} \tag{2}$$

is orthogonal to the marginals $\eta_1$ and $\eta_2$.

We have another interpretation of $\theta$. Let's expand $p(\boldsymbol{x})$ by $\log p(\boldsymbol{x}) = \sum_{i=1}^{2} \theta_i x_i + \theta_{12} x_1 x_2 - \psi$. Simple calculation lets us get the coefficients, $\theta_1 = \log \frac{p_{10}}{p_{00}}$, $\theta_2 = \log \frac{p_{01}}{p_{00}}$, $\psi = -\log p_{00}$, and $\theta = \theta_{12}$ (as Eq 2). The triplet $\boldsymbol{\theta} = (\theta_1, \theta_2, \theta_{12})$ forms another coordinate system, called the natural parameters, or $\theta$−coordinates. We remark that $\theta_{12}$ is 0 when and only when $X_1$ and $X_2$ are independent.

The triplet

$$\boldsymbol{\zeta} \equiv (\eta_1, \eta_2, \theta_{12})$$

forms an 'orthogonal' coordinate system of $\boldsymbol{S}_2$, called the mixed coordinates [4].

We use the Kullback-Leibler divergence (KL) to measure the discrepancy between two probabilities $p(\boldsymbol{x})$ and $q(\boldsymbol{x})$, defined by $D[p : q] = \sum_{\boldsymbol{x}} p(\boldsymbol{x}) \log \frac{p(\boldsymbol{x})}{q(\boldsymbol{x})}$. In the following, we denote any coordinates of $p$ by $\boldsymbol{\xi}^p$ etc (the same for $q$). Using the orthogonality between $\eta$- and $\theta$-coordinates, we have the decomposition in the KL.

**Theorem 2.**

$$D[p : q] = D[p : r^*] + D[r^* : q], \qquad D[q : p] = D[q : r^{**}] + D[r^{**} : p], \tag{3}$$

where $r^*$ and $r^{**}$ are given by $\boldsymbol{\zeta}^{r^*} = (\eta_1^p, \eta_2^p, \theta_3^q)$ and $\boldsymbol{\zeta}^{r^{**}} = (\eta_1^q, \eta_2^q, \theta_3^p)$, respectively.

The squared distance $ds^2$ between two nearby distributions $p(\boldsymbol{x}, \boldsymbol{\xi})$ and $p(\boldsymbol{x}, \boldsymbol{\xi}, +d\boldsymbol{\xi})$ is given by the quadratic form of $d\boldsymbol{\xi}$,

$$ds^2 = \sum_{i,j \in (1,2,3)} g_{ij}(\boldsymbol{\xi}) d\xi_i d\xi_j,$$

which is approximately twice the KL, i.e., $ds^2 \approx 2D[p(\boldsymbol{x}, \boldsymbol{\xi}) : p(\boldsymbol{x}, \boldsymbol{\xi} + d\boldsymbol{\xi})]$.

Now suppose $\boldsymbol{\xi}$ is the mixed coordinates $\boldsymbol{\zeta}$. Then, the Fisher information matrix is of the form $g_{ij}^{\zeta} = \begin{bmatrix} g_{11}^{\zeta} & g_{12}^{\zeta} & 0 \\ g_{12}^{\zeta} & g_{22}^{\zeta} & 0 \\ 0 & 0 & g_{33}^{\zeta} \end{bmatrix}$ and we have $ds^2 = ds_1^2 + ds_2^2$, where $ds_1^2 = g_{33}^{\zeta}(d\theta_3)^2$, $ds_2^2 = \sum_{i,j \in (1,2)} g_{ij}^{\zeta} d\eta_i d\eta_j$, corresponding to Eq. 3.

This decomposition comes from the choice of the orthogonal coordinates and gives us the merits of simple procedure in statistical inference. First, let us estimate the parameter $\boldsymbol{\eta} = (\eta_1, \eta_2)$ and $\theta$ from $N$ observed data $\boldsymbol{x}_1, ..., \boldsymbol{x}_N$. The maximum likelihood estimator (mle) $\hat{\boldsymbol{\zeta}}$, which is asymptotically unbiased and efficient, is easily obtained by $\hat{\eta}_i = \frac{1}{N}\#\{x_i = 1\}$ and $\hat{\theta} = \log \frac{\hat{\eta}_{12}(1-\hat{\eta}_1-\hat{\eta}_2+\hat{\eta}_{12})}{(\hat{\eta}_1-\hat{\eta}_{12})(\hat{\eta}_2-\hat{\eta}_{12})}$, using $\hat{\eta}_{12} = \frac{1}{N}\#\{x_1 x_2 = 1\}$. The covariance of estimation error, $\Delta\boldsymbol{\eta}$ and $\Delta\theta$, is given asymptotically by $\mathrm{Cov}\begin{bmatrix} \Delta\boldsymbol{\eta} \\ \Delta\theta \end{bmatrix} = \frac{1}{N}G_{\zeta}^{-1}$. Since the cross terms of $G$ or $G^{-1}$ vanish for the orthogonal coordinates, we have $\mathrm{Cov}\,[\Delta\boldsymbol{\eta}, \Delta\theta] = 0$, implying that the estimation error $\Delta\boldsymbol{\eta}$ of marginals and that of interaction are mutually independent. Such a property does not hold for other non-orthogonal parameterization such as the COR $\rho$, the covariance etc. Second, in practice, we often like to compare many spike distributions, $q(\boldsymbol{x}(t))$ (i.e, $\boldsymbol{\zeta}^{q(t)}$) for $(t = 1,,,T)$, with a distribution in the control period $p(\boldsymbol{x})$, or $\boldsymbol{\zeta}^p$. Because the orthogonality between $\boldsymbol{\eta}$ and $\theta$ allows us to treat them independently, these comparisons become very simple.

These properties bring a simple procedure of testing hypothesis concerning the null hypothesis
$$H_0 : \theta = \theta_0 \qquad \text{against} \qquad H_1 : \theta \neq \theta_0, \qquad (4)$$
where $\theta_0$ is not necessarily zero, whereas $\theta_0 = 0$ corresponds to the null hypothesis of independent firing, which is often used in literature in different setting. Let the log likelihood of the models $H_0$ and $H_1$ be, respectively,
$$l_0 = \max_{\boldsymbol{\eta}} \log p(\boldsymbol{x}_1, ..., \boldsymbol{x}_N; \boldsymbol{\eta}, \theta_0) \text{ and } l_1 = \max_{\boldsymbol{\eta}, \theta} \log p(\boldsymbol{x}_1, ..., \boldsymbol{x}_N; \boldsymbol{\eta}, \theta).$$

The likelihood ratio test uses the test statistics $\lambda = 2\log\frac{l_0}{l_1}$. By the mle with respect to $\boldsymbol{\eta}$ and $\theta$, which can be performed independently, we have
$$l_0 = \log p(\bar{\boldsymbol{x}}, \hat{\boldsymbol{\eta}}, \theta_0), \qquad l_1 = \log p(\bar{\boldsymbol{x}}, \hat{\boldsymbol{\eta}}, \hat{\eta}_{12}), \qquad (5)$$
where $\hat{\boldsymbol{\eta}}$ are the same in both models. A similar situation holds in the case of testing $\boldsymbol{\eta} = \boldsymbol{\eta}_0$ against $\boldsymbol{\eta} \neq \boldsymbol{\eta}_0$ for unknown $\theta$.

Under the hypothesis $H_0$, $\lambda$ is approximated for a large $N$ as
$$\lambda = 2\sum_{i=1}^{N} \log \frac{p(\boldsymbol{x}_i; \hat{\boldsymbol{\eta}}, \theta_0)}{p(\boldsymbol{x}_i; \hat{\boldsymbol{\eta}}, \hat{\theta})} \approx N g_{33}^{\zeta}(\hat{\theta} - \theta_0)^2 \sim \chi^2(1). \qquad (6)$$

Thus, we can easily submit our data to a hypothetical testing of significant coincident firing against null hypothesis of any correlated firing, independently from the mean firing rate modulation[1] .

We now turn to relate the above approach with another important issue, which is to relate such a coincident firing with behavior. Let us denote by Y a variable of discrete behavioral choices. The MI between $X = (X_1, X_2)$ and $Y$ is written by
$$I(X, Y) = E_{p(X,Y)}\left[\log \frac{p(\boldsymbol{x}, y)}{p(\boldsymbol{x})p(y)}\right] = E_{p(Y)}\left[D\left[p(X|y) : p(X)\right]\right].$$

Using the mixed coordinates for $p(X|y)$ and $p(X)$, we have $D\left[p(X|y) : p(X)\right] = D\left[\boldsymbol{\zeta}(X|y) : \boldsymbol{\zeta}(X)\right] = D\left[\boldsymbol{\zeta}(X|y) : \boldsymbol{\zeta}'\right] + D\left[\boldsymbol{\zeta}' : \boldsymbol{\zeta}(X)\right]$, where $\boldsymbol{\zeta}' = \boldsymbol{\zeta}'(X, y) = (\zeta_1(X|y), \zeta_2(X|y), \zeta_3(X)) = (\eta_1(X|y), \eta_2(X|y), \theta_3(X))$.

**Theorem 3.**

$$I(X, Y) = I_1(X, Y) + I_2(X, Y), \tag{7}$$

where $I_1(X, Y), I_2(X, Y)$ are given by

$$I_1(X, Y) = E_{p(Y)} \left[ D \left[ \zeta(X|y) : \zeta'(X, y) \right] \right], I_2(X, Y) = E_{p(Y)} \left[ D \left[ \zeta'(X, y) : \zeta(X) \right] \right].$$

Obviously, the similar result holds with respect to $p(Y|X)$. By this theorem, $I$ is the sum of the two terms: $I_1$ is by modulation of the correlation components of $X$, while $I_2$ is by modulation of the marginals of $X$. This observation helps us investigate the behavioral significance by modulating either coincident firing or mean firing rates.

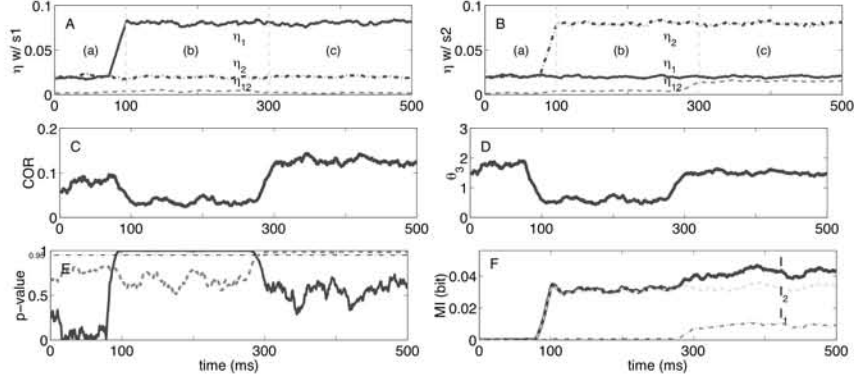

Figure 1: Demonstration of information-geometric measure in two neuron case, using simulated neural data, where two behavioral choices (s1, s2) are assumed. A,B. $(\eta_1, \eta_2, \eta_{12})$ with respect to s1, s2. C,D. COR,$\theta$, computed by using $\boldsymbol{\eta} = \sum_i p(s_i) \boldsymbol{\eta}(s_i)$ with $p(s_i) = 1/2$ ($i = 1, 2$). E. p-values. F. MI.

Fig 1 succinctly demonstrates results in this section. Figs 1 A, B are supposed to show mean firing rates of two neurons and mean coincident firing for two different stimuli (s1, s2). The period (a) is assumed as the control period, i.e., where no stimuli is shown yet, whereas the stimulus is shown in the periods (b,c). Fig 1 C, D gives COR, $\theta$. They look to change similarly over periods, which is reasonable because both COR and $\theta$ represent the same correlational component, but indeed change slightly differently over periods (e.g., the relative magnitudes between the periods (a) and (c) are different for COR and $\theta$), which is also reasonable because both represent the correlational component as in different coordinate systems. Using $\theta$ in Fig 1 D, Fig 1 E shows p-values derived from $\chi^2(1)$ (i.e., $p > 0.95$ in Fig 1 E is 'a significance with $p < 0.05$') for two different null hypotheses, one of the averaged firing in the control period (by solid line) and the other of independent firing (by dashed line), which is of popular use in literature.

In general, it becomes complicated to test the former hypothesis, using COR. This is because the COR, as the coordinate component, is not orthogonal to the mean firing rates so that estimation errors among the COR and mean firing rates are entangled and that the proper metric among them is rather difficult to compute. Once using $\theta$, this testing becomes simple due to orthogonality between $\theta$ and mean firing rates.

Notably, we would draw completely different conclusions on significant coincident firing given each null hypothesis in Fig 1 E. This difference may be striking when we are to understand the brain function with these kinds of data. Fig 1 F shows the MI

between firing and behavior, where behavioral event is with respect to stimuli, and its decomposition. There is no behavioral information conveyed by the modulation of coincident firing in the period (b) (i.e., $I_1 = 0$ in the period (b)). The increase in the total MI (i.e., $I$) in the period (c), compared with the period (b), is due not to the MI in mean firing ($I_2$) but to the MI correlation ($I_1$). Thus, with a great ease, we can directly inspect a function of neural correlation component in relation to behavior.

## 3 Three neuron case

With more than two neurons, we need to look not only into a pairwise interaction but also into higher-order interactions. Our results in the two neuron case are naturally extended to $n$ neuron case and here, we focus on three neuron case for illustration.

For three neurons $X = (X_1, X_2, X_3)$, we let $p(\boldsymbol{x})$, $\boldsymbol{x} = (x_1, x_2, x_3)$, be their joint probability distribution and put $p_{ijk} = \mathrm{Prob}\{x_1 = i, x_2 = j, x_3 = k\}$, $i, j, k = 0, 1$. The set of all such distributions forms a 7-dimensional manifold $\boldsymbol{S}_3$ due to $\sum p_{ijk} = 1$. The $\eta$-coordinates $\boldsymbol{\eta} = (\boldsymbol{\eta}_1; \boldsymbol{\eta}_2; \boldsymbol{\eta}_3) = (\eta_1, \eta_2, \eta_3; \eta_{12}, \eta_{23}, \eta_{13}; \eta_{123})$ is defined by

$$\eta_i = E[x_i] \quad (i = 1, 2, 3), \quad \eta_{ij} = E[x_i x_j] \quad (i, j = 1, 2, 3; i \neq j), \quad \eta_{123} = E[x_1 x_2 x_3].$$

To single out the purely triplewise correlation, we utilize the dual orthogonality of $\theta$- and $\eta$-coordinates. By using expansion of $\log p(\boldsymbol{x}) = \sum \theta_i x_i + \sum \theta_{ij} x_i x_j + \theta_{123} x_1 x_2 x_3 - \psi$, we obtain $\theta$-coordinates, $\boldsymbol{\theta} = (\boldsymbol{\theta}_1; \boldsymbol{\theta}_2; \boldsymbol{\theta}_3) = (\theta_1, \theta_2, \theta_3; \theta_{12}, \theta_{23}, \theta_{13}; \theta_{123})$. It's easy to get the expression of these coefficients (e.g., $\theta_{123} = \log \frac{p_{111} p_{100} p_{010} p_{001}}{p_{110} p_{101} p_{011} p_{000}}$). Information geometry gives the following theorem.

**Theorem 4.** $\theta_{123}$ represents the pure triplewise interaction in the sense that it is orthogonal to any changes in the single and pairwise marginals, i.e., $\boldsymbol{\eta}_1$ and $\boldsymbol{\eta}_2$.

We use the following two mixed coordinates to utilize the dual orthogonality,

$$\boldsymbol{\zeta}_1 = (\boldsymbol{\eta}_1; \boldsymbol{\theta}_2; \boldsymbol{\theta}_3), \boldsymbol{\zeta}_2 = (\boldsymbol{\eta}_1; \boldsymbol{\eta}_2; \boldsymbol{\theta}_3).$$

Here $\boldsymbol{\zeta}_2$ is useful to single out the triplewise interaction ($\boldsymbol{\theta}_3 = \theta_{123}$), while $\boldsymbol{\zeta}_1$ is to single out the pairwise and triplewise interactions together ($\boldsymbol{\theta}_2; \boldsymbol{\theta}_3$). Note that $\theta_{123}$ is not orthogonal to $\{\theta_{ij}\}$. In other words, except the case of no triplewise interaction ($\theta_{123} = 0$), $\theta_{ij}$ do not directly represent the pairwise correlation of two random variables $X_i, X_j$. The case of independent firing is given by $\eta_{ij} = \eta_i \eta_j, \eta_{123} = \eta_1 \eta_2 \eta_3$ or equivalently by $\boldsymbol{\theta}_2 = 0, \boldsymbol{\theta}_3 = 0$.

The decomposition in the KL is now given as follows.

**Theorem 5.**

$$D[p:q] = D[p:\bar{p}] + D[\bar{p}:q] = D[p:\tilde{p}] + D[\tilde{p}:q] = D[p:\bar{p}] + D[\bar{p}:\tilde{p}] + D[\tilde{p}:q]. \tag{8}$$

where, using the mixed coordinates, we have $\boldsymbol{\zeta}_2^{\bar{p}} = (\boldsymbol{\eta}_1^p; \boldsymbol{\eta}_2^p; \boldsymbol{\theta}_3^q), \boldsymbol{\zeta}_1^{\tilde{p}} = (\boldsymbol{\eta}_1^p; \boldsymbol{\theta}_2^q; \boldsymbol{\theta}_3^q)$.

A hypothetical testing is formulated similarly to the two neuron case. We can examine a significance of the triplewise interaction by $\lambda_2 = 2ND[p:\bar{p}] \approx N g_{77}^{\zeta}(\boldsymbol{\zeta}_2^p)(\theta_{123}^p - \theta_{123}^q)^2 \sim \chi^2(1)$. For a significance of triplewise and pairwise interactions together, we have $\lambda_1 = 2ND[p:\tilde{p}] \approx N \sum_{i,j=4}^{7} g_{ij}^{\zeta}(\boldsymbol{\zeta}_1^p)(\zeta_i^p - \zeta_i^{\tilde{p}})(\zeta_j^p - \zeta_j^{\tilde{p}}) \sim \chi^2(4)$.

For the decomposition of the MI between firing $X$ and behavior $Y$, we have

**Theorem 6.**

$$I(X, Y) = I_1(X, Y) + I_2(X, Y) = I_3(X, Y) + I_4(X, Y) \tag{9}$$

where

$$I_1(X,Y) = E_{p(Y)}\left[D\left[\zeta_1(X|y):\zeta_1(X,y)\right]\right], \ I_2(X,Y) = E_{p(Y)}\left[D\left[\zeta_1(X,y):\zeta_1(X)\right]\right],$$

$$I_3(X,Y) = E_{p(Y)}\left[D\left[\zeta_2(X|y):\zeta_2(X,y)\right]\right], \ I_4(X,Y) = E_{p(Y)}\left[D\left[\zeta_2(X,y):\zeta_2(X)\right]\right].$$

By the first equality, $I$ is decomposed into two parts: $I_1$ is conveyed by the pairwise and triplewise interactions of firing, and $I_2$ by the mean firing rate modulation. By the second equality, $I$ is decomposed differently: $I_3$, conveyed by the triplewise interaction, and $I_4$, by the other terms.

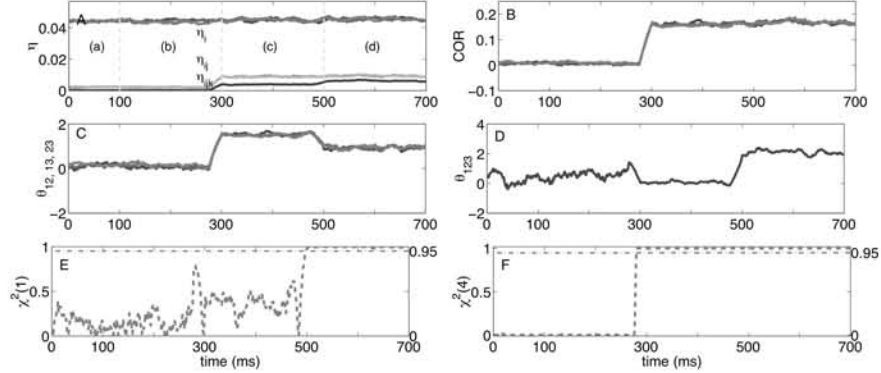

Figure 2: Demonstration in three neuron case. A $\boldsymbol{\eta} = (\boldsymbol{\eta}_1, \boldsymbol{\eta}_2, \boldsymbol{\eta}_3) \approx (\eta_i, \eta_{ij}, \eta_{ijk})$ from top to bottom, since we treated a homogeneous case in this simulation for simplicity. B. COR. C. $\theta_{12}, \theta_{13}, \theta_{23}$. D $\theta_{123}$. E p-value $\sim \chi^2(1)$. F p-value $\sim \chi^2(4)$.

We emphasize that all the above decompositions come from the choice of the 'orthogonal' coordinates. Fig 2 highlights some of the results in this section. Fig 2 A shows the mean firing rates (see legend). The period (a) is assumed as the control period. Fig 2 B indicates that COR changes only in the periods (c,d), while Fig 2 C indicates that $\theta_{123}$ changes only in the period (d). Taken together, we observe that the triplewise correlation $\theta_{123}$ can be modulated independently from COR. Fig 2 E indicates the p-value from $\chi^2(1)$ against the null hypothesis of the activity in the control period. The triplewise coincident firing becomes significant only in the period (d). Fig 2 F indicates the p-value from $\chi^2(4)$. The coincident firing, taking the triplewise and pairwise interaction together, becomes significant in both periods (c,d). We cannot observe these differences in modulation of pairwise and triplewise interactions over periods (c, d), when we inspect only COR.

**Remark**: For a general $n$ neuron case, we can use the $k$-cut mixed coordinates, $\zeta_k = (\boldsymbol{\eta}_1, ..., \boldsymbol{\eta}_k; \boldsymbol{\theta}_{k+1}, ..., \boldsymbol{\theta}_n) = (\boldsymbol{\eta}_{k-}; \boldsymbol{\theta}_{k+})$. Using the orthogonality between $\boldsymbol{\eta}_{k-}$ and $\boldsymbol{\theta}_{k+}$, the similar results hold. To meet the computational complexity involved in this general case, some practical difficulties should be resolved in practice [9].

## 4    Discussions

We presented the information-geometric measures to analyze spike firing patterns, using two and three neuron cases for illustration. The choice of 'orthogonal' coordinates provides us with a simple, transparent and systematic procedure to test significant firing patterns and to directly relate such a pattern with behavior. We hope that this method simplifies and strengthens experimental data analysis.

## Acknowledgments

HN thanks M. Tatsuno, K. Siu and K. Kobayashi for their assistance. HN is supported by Grants-in-Aid 13210154 from the Ministry of Edu. Japan.

## Footnotes

[1] A more proper formulation in this hypothetical testing can be derived, resulting in using $p$ value from $\chi^2(2)$ distribution, but we omit it here due to the limited space [9]

## References

[1] M. Abeles, H. Bergman, E. Margalit, and E. Vaadia. Spatiotemporal firing patterns in the frontal cortex of behaving monkeys. *J Neurophysiol*, 70(4):1629–38., 1993.

[2] A. M. H. J. Aertsen, G. L. Gerstein, M. K. Habib, and G. Palm. Dynamics of neuronal firing correlation: Modulation of "effective connectivity". *Journal of Neurophysiology*, 61(5):900–917, May 1989.

[3] S. Amari. Information geometry on hierarchical decomposition of stochastic interactions. *IEEE Transaction on Information Theory*, pages 1701–1711, 2001.

[4] S. Amari and H. Nagaoka. *Methods of Information Geometry*. AMS and Oxford University Press, 2000.

[5] S. Grün. *Unitary joint-events in multiple-neuron spiking activity: detection, significance, and interpretation*. Verlag Harri Deutsch, Reihe Physik, Band 60. Thun, Frankfurt/Main, 1996.

[6] H. Ito and S. Tsuji. Model dependence in quantification of spike interdependence by joint peri-stimulus time histogram. *Neural Computation*, 12:195–217, 2000.

[7] L. Martignon, G. Deco, K. Laskey, M. Diamond, W. A. Freiwald, and E. Vaadia. Neural coding: Higher-order temporal patterns in the neurostatistics of cell assemblies. *Neural Computation*, 12(11):2621–2653, 2000.

[8] H. Nagaoka and S. Amari. Differential geometry of smooth families of probability distributions. Technical report, University of Tokyo, 1982.

[9] H. Nakahara and S. Amari. Information geometric measure for neural spikes. in prepration.

[10] H. Nakahara, S. Amari, M. Tatsuno, S. Kang, K. Kobayashi, K. Anderson, E. Miller, and T. Poggio. Information geometric measures for spike firing. *Society for Neuroscience Abstracts*, 27:821.46 (page.2178), 2001.

[11] M. W. Oram, N. G. Hatsopoulos, B. J. Richmond, and J. P. Donoghue. Excess synchrony in motor cortical neurons provides redundant direction information with that from coarse temporal measures. *J Neurophysiol.*, 86(4):1700–1716, 2001.

[12] S. Panzeri and S. R. Schultz. A unified approach to the study of temporal, correlational, and rate coding. *Neural Computation*, 13(6):1311–49., 2001a.

[13] A. Riehle, S. Grün, M. Diesmann, and A. Aertsen. Spike synchronization and rate modulation differentially involved in motor cortical function. *Science*, 278:1950–1953, 12 Dec 1997.

[14] E. Vaadia, I. Haalman, M. Abeles, H. Bergman, Y. Prut, H. Slovin, and A. Aertsen. Dynamics of neuronal interactions in monkey cortex in relation to behavioural events. *Nature*, 373:515–518, 9 Feb 1995.
